# Probabilistic Anomaly Detection in Dynamic Systems

**Padhraic Smyth**
Jet Propulsion Laboratory 238-420
California Institute of Technology
4800 Oak Grove Drive
Pasadena, CA 91109

## Abstract

This paper describes probabilistic methods for novelty detection when using pattern recognition methods for fault monitoring of dynamic systems. The problem of novelty detection is particularly acute when prior knowledge and training data only allow one to construct an incomplete classification model. Allowance must be made in model design so that the classifier will be robust to data generated by classes not included in the training phase. For diagnosis applications one practical approach is to construct both an input density model and a discriminative class model. Using Bayes' rule and prior estimates of the relative likelihood of data of known and unknown origin the resulting classification equations are straightforward. The paper describes the application of this method in the context of hidden Markov models for online fault monitoring of large ground antennas for spacecraft tracking, with particular application to the detection of transient behaviour of unknown origin.

## 1  PROBLEM BACKGROUND

Conventional control-theoretic models for fault detection typically rely on an accurate model of the plant being monitored (Patton, Frank, and Clark, 1989). However, in practice it common that no such model exists for complex non-linear systems. The large ground antennas used by JPL's Deep Space Network (DSN) to track

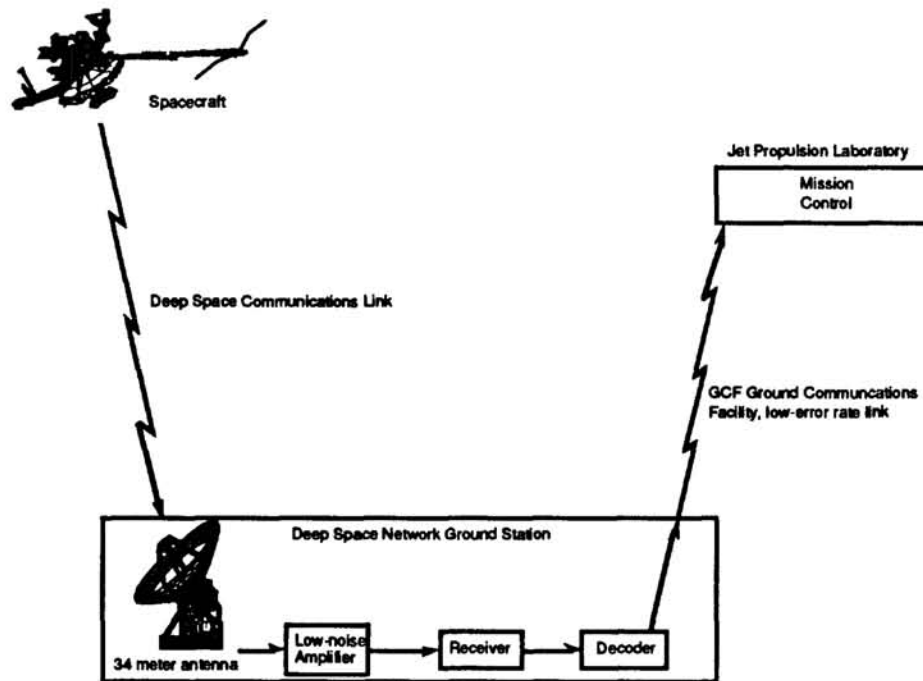

Figure 1: Block diagram of typical Deep Space Network downlink

planetary spacecraft fall into this category. Quite detailed analytical models exist for the electromechanical pointing systems. However, these models are primarily used for determining gross system characteristics such as resonant frequencies; they are known to be a poor fit for fault detection purposes.

We have previously described the application of adaptive pattern recognition methods to the problem of online health monitoring of DSN antennas (Smyth and Mellstrom, 1992; Smyth, in press). Rapid detection and identification of failures in the electromechanical antenna pointing systems is highly desirable in order to minimize antenna downtime and thus minimise telemetry data loss when communicating with remote spacecraft (see Figure 1). Fault detection based on manual monitoring of the various antenna sensors is neither reliable or cost-effective.

The pattern-recognition monitoring system operates as follows. Sensor data such as motor current, position encoder, tachometer voltages, and so forth are synchronously sampled at 50Hz by a data acquisition system. The data are blocked off into disjoint windows (200 samples are used in practice) and various features (such as estimated autoregressive coefficients) are extracted; let the feature vector be $\underline{\theta}$.

The features are fed into a classification model (every 4 seconds) which in turn provides posterior probability estimates of the $m$ possible states of the system given the estimated features from that window, $p(\omega_i|\underline{\theta})$. $\omega_1$ corresponds to normal conditions, the other $\omega_i$'s, $1 \leq i \leq m$, correspond to known fault conditions.

Finally, since the system has "memory" in the sense that it is more likely to remain in the current state than to change states, the posterior probabilities need to be correlated over time. This is achieved by a standard first-order hidden Markov

model (HMM) which models the temporal state dependence. The hidden aspect of the model reflects the fact that while the features are directly observable, the underlying system states are not, i.e., they are in effect "hidden." Hence, the purpose of the HMM is to provide a model from which the most likely sequence of system states can be inferred *given* the observed sequence of feature data.

The classifier portion of the model is trained using simulated hardware faults. The feed-forward neural network has been the model of choice for this application because of its discrimination ability, its posterior probability estimation properties (Richard and Lippmann, 1992; Miller, Goodman and Smyth, 1993) and its relatively simple implementation in software. It should be noted that unlike typical speech recognition HMM applications, the transition probabilities are not estimated from data but are designed into the system based on prior knowledge of the system mean time between failure (MTBF) and other specific knowledge of the system configuration (Smyth, in press).

## 2    LIMITATIONS OF THE DISCRIMINATIVE MODEL

The model described above assumes that there are $m$ known mutually exclusive and exhaustive states (or "classes") of the system, $\omega_1, \ldots, \omega_m$. The mutually exclusive assumption is reasonable in many applications where multiple simultaneous failures are highly unlikely. However, the exhaustive assumption is somewhat impractical. In particular, for fault detection in a complex system such as a large antenna, there are thousands of possible fault conditions which might occur. The probability of occurrence of any single condition is very small, but nonetheless there is a significant probability that at least one of these conditions will occur over some finite time. While the common faults can be directly modelled it is not practical to assign model states to all the other minor faults which might occur.

As discussed in (Smyth and Mellstrom, 1992; Smyth 1994) a discriminative model directly models $p(\omega_i|\underline{\theta})$, the posterior probabilities of the classes given the feature data, and assumes that the classes $\omega_1, \ldots, \omega_m$ are exhaustive. On the other hand, a *generative* model directly models the probability density function of the input data conditioned on each class, $p(\underline{\theta}|\omega_i)$, and then indirectly determines posterior class probabilities by application of Bayes' rule. Examples of generative classifiers include parametric models such as Gaussian classifiers and memory-based methods such as kernel density estimators. Generative models are by nature well suited to novelty detection whereas discriminative models have no built-in mechanism for detecting data which are different to that on which the model was trained. However, there is a trade-off; because generative models typically are doing more modelling than just searching for a decision boundary, they can be less efficient (than discriminant methods) in their use of the data. For example, generative models typically scale poorly with input dimensionality for fixed training sample size.

## 3    HYBRID MODELS

A relatively simple and practical approach to the novelty detection problem is to use *both* a generative and discriminative classifier (an idea originally suggested to the author by R. P. Lippmann). An extra "$m+1$th" state is added to the model to

cover "all other possible states" not accounted for by the known $m$ states. In this framework, the posterior estimates of the discriminative classifier are conditioned on the event that the data come from one of the $m$ known classes.

Let the symbol $\omega_{\{1,...,m\}}$ denote the event that the true system state is one of the known states, let $\omega_{m+1}$ be the unknown state, and let $p(\omega_{m+1}|\underline{\theta})$ be the posterior probability that the system is in an unknown state given the data. Hence, one can estimate the posterior probability of individual known states as

$$\hat{p}(\omega_i|\underline{\theta}) = p_d(\omega_i|\underline{\theta}, \omega_{\{1,...,m\}}) \times (1 - p(\omega_{m+1}|\underline{\theta})), \qquad 1 \leq i \leq m \qquad (1)$$

where $p_d(\omega_i|\underline{\theta}, \omega_{\{1,...,m\}})$ is the posterior probability estimate of state $i$ as provided by a discriminative model, i.e., given that the system is in one of the known states.

The calculation of $p(\omega_{m+1}|\underline{\theta})$ can be obtained via the usual application of Bayes' rule if $p(\underline{\theta}|\omega_{m+1})$, $p(\omega_{m+1})$, and $p(\underline{\theta}|\omega_{\{1,...,m\}})$ are known:

$$p(\omega_{m+1}|\underline{\theta}) = \frac{p(\underline{\theta}|\omega_{m+1})p(\omega_{m+1})}{p(\underline{\theta}|\omega_{m+1})p(\omega_{m+1}) + p(\underline{\theta}|\omega_{\{1,...,m\}})\sum_i^m p(\omega_i)}. \qquad (2)$$

Specifying the prior density $p(\underline{\theta}|\omega_{m+1})$, the distribution of the features conditioned on the occurrence of the unknown state, can be problematic. In practice we have used non-informative Bayesian priors for $p(\underline{\theta}|\omega_{m+1})$ over a bounded space of feature values (details are available in a technical report (Smyth and Mellstrom, 1993)), although the choosing of a prior density for data of unknown origin is basically ill-posed. The stronger the constraints which can be placed on the features the narrower the resulting prior density and the better the ability of the overall model to detect novelty. If we only have very weak prior information, this will translate into a weaker criterion for accepting points which belong to the unknown category.

The term $p(\omega_{m+1})$ (in Equation (2)) must be chosen based on the designer's prior belief of how often the system will be in an unknown state — a practical choice is that the system is at least as likely to be in an unknown failure state as any of the known failure states.

The $p(\underline{\theta}|\omega_{\{1,...,m\}})$ term in Equation (2) is provided directly by the generative model. Typically this can be a mixture of Gaussian component densities or a kernel density estimate over all of the training data (ignoring class labels). In practice, for simplicity of implementation we use a simple Gaussian mixture model. Furthermore, because of the afore-mentioned scaling problem with input dimensions, only a subset of relatively significant input features are used in the mixture model. A less heuristic approach to this aspect of the problem (with which we have not yet experimented) would be to use a method such as projection pursuit to project the data into a lower dimensional subspace and perform the input density estimation in this space. The main point is that the generative model need not necessarily work in the full dimensional space of the input features.

Integration of Equations (1) and (2) into the hidden Markov model scheme is straightforward and is not derived here — the HMM now has an extra state, "unknown." The choice of transition probabilities between the unknown and other states is once again a matter of design choice. For the antenna application at least, many of the unknown states are believed to be relatively brief transient phenomena which

last perhaps no longer than a few seconds: hence, the Markov matrix is designed to reflect these beliefs since the expected duration of any state $d[\omega_i]$ (in units of sampling intervals) must obey

$$d[\omega_i] = \frac{1}{1 - p_{ii}} \tag{3}$$

where $p_{ii}$ is the self-transition probability of state $\omega_i$.

## 4   EXPERIMENTAL RESULTS

For illustrative purposes the experimental results from 2 particular models are compared. Each was applied to monitoring the servo pointing system of a DSN 34m antenna at Goldstone, California. The models were implemented within LabView data acquisition software running in real-time on a Macintosh II computer at the antenna site. The models had previously been trained off-line on data collected some months earlier. 12 input features were used consisting of estimated autoregressive coefficients and variance terms from each window of 200 samples of multichannel data. For both models a discriminative feedforward neural network model (with 8 hidden units, sigmoidal hidden and output activation functions) was trained (using conjugate-gradient optimization) to discriminate between a normal state and 3 known and commonly occurring fault states (failed tachometer, noisy tachometer, and amplifier short circuit — also known as "compensation loss"). The network output activations were normalised to sum to 1 in order to provide posterior class probability estimates.

Model (a) used no HMM and assumed that the 4 known states are exhaustive, i.e., it just used the feedforward network. Model (b) used a HMM with 5 states, where a generative model (a Gaussian mixture model) and a flat prior (with bounds on the feature values) were used to determine the probability of the 5th state (as described by Equations (1) and (2)). The same neural network as in model (a) was used as a discriminator for the other 4 known states. The generative mixture model had 10 components and used only 2 of the 12 input features, the 2 which were judged to be the most sensitive to system change. The parameters of the HMM were designed according to the guidelines described earlier. Known fault states were assumed to be equally likely with 1 hour MTBF's and with 1 hour mean duration. Unknown faults were assumed to have a 20 minute MTBF and a 10 second mean duration. Both HMMs used 5-step backwards smoothing, i.e., the probability estimates at any time $n$ are based on all past data up to time $n$ and future data up to time $n + 5$ (using a larger number of backward steps was found empirically to produce no effect on the estimates).

Figures 2 (a) and (b) show each model's estimates (as a function of time) that the system is in the normal state. The experiment consisted of introducing known hardware faults into the system in a controlled manner after 15 minutes and 45 minutes, each of 15 minutes duration.

Model (a)'s estimates are quite noisy and contain a significant number of potential false alarms (highly undesirable in an operational environment). Model (b) is much more stable due to the smoothing effect of the HMM. Nonetheless, we note that between the 8th and 10th minutes, there appear to be some possible false alarms:

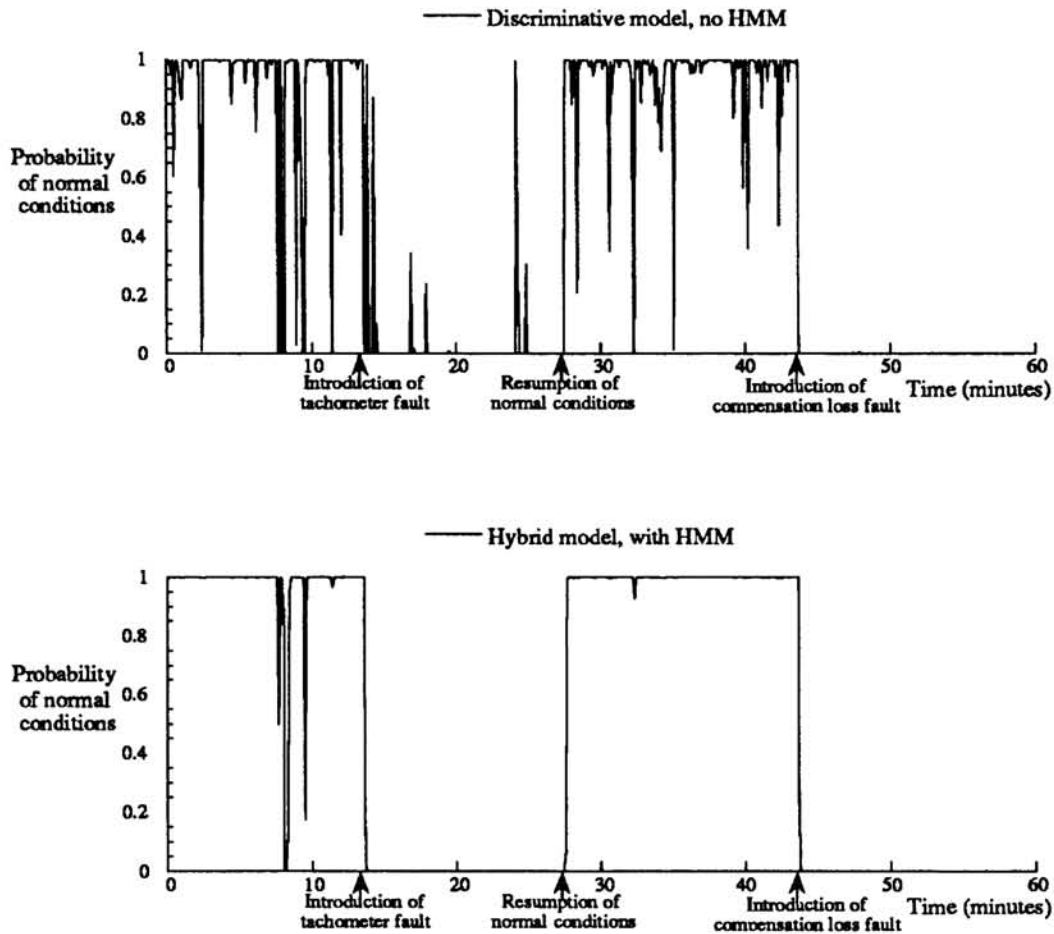

Figure 2: Estimated posterior probability of normal state (a) using no HMM and the exhaustive assumption (normal + 3 fault states), (b) using a HMM with a hybrid model (normal + 3 faults + other state).

these data were classified into the unknown state (not shown). On later inspection it was found that large transients (of unknown origin) were in fact present in the original sensor data and that this was what the model had detected, confirming the classification provided by the model. It is worth pointing out that the model without a generative component (whether with or without the HMM) also detected a non-normal state at the same time, but incorrectly classified this state as one of the known fault states (these results are not shown).

Also not shown are the results from using a generative model alone, with no discriminative component. While its ability to detect unknown states was similar to the hybrid model, its ability to discriminate between known states was significantly worse than the hybrid model.

The hybrid model has been empirically tested on a variety of other conditions where various "known" faults are omitted from the discriminative training step and then

presented to the model during testing: in all cases, the anomalous unknown state was detected by the model, i.e., classified as a state which the model had not seen before.

# 5    APPLICATION ISSUES

The model described here is currently being integrated into an interactive antenna health monitoring software tool for use by operations personnel at all new DSN antennas. The first such antenna is currently being built at the Goldstone (California) DSN site and is scheduled for delivery to DSN operations in late 1994. Similar antennas, also equipped with fault detectors of the general nature described here, will be constructed at the DSN ground station complexes in Spain and Australia in the 1995-96 time-frame.

The ability to detect previously unseen transient behaviour has important practical consequences: as well as being used to warn operators of servo problems in real-time, the model will also be used as a filter to a data logger to record interesting and anomalous servo data on a continuous basis. Hence, potentially novel system characteristics can be recorded for correlation with other antenna-related events (such as maser problems, receiver lock drop during RF feedback tracking, etc.) for later analysis to uncover the true cause of the anomaly. A long-term goal is to develop an algorithm which can automatically analyse the data which have been classified into the unknown state and extract distinct sub-classes which can be added as new explicit states to the HMM monitoring system in a dynamic fashion. Stolcke and Omohundro (1993) have described an algorithm which dynamically creates a state model for HMMs for the case of discrete-valued features. The case of continuous-valued features is considerably more subtle and may not be solvable unless one makes significant prior assumptions regarding the nature of the data-generating mechanism.

# 6    CONCLUSION

A simple hybrid classifier was proposed for novelty detection within a probabilistic framework. Although presented in the context of hidden Markov models for fault detection, the proposed scheme is perfectly general for generic classification applications. For example, it would seem highly desirable that fielded automated medical diagnosis systems (such as various neural network models which have been proposed in the literature) should always contain a "novelty-detection" component in order that novel data are identified and appropriately classified by the system.

The primary weakness of the methodology proposed in this paper is the necessity for prior knowledge in the form of densities for the feature values given the unknown state. The alternative approach is not to explicitly model the the data from the unknown state but to use some form of thresholding on the input densities from the known states (Aitchison, Habbema, and Kay, 1977; Dubuisson and Masson, 1993). However, direct specification of threshold levels is itself problematic. In this sense, the specification of prior densities can be viewed as a method for automatically determining the appropriate thresholds (via Equation (2)).

As a final general comment, it is worth noting that *online* learning systems must use some form of novelty detection. Hence, hybrid generative-discriminative models (a simple form of which has been proposed here) may be a useful framework for modelling online learning.

## Acknowledgements

The author would like to thank Jeff Mellstrom, Paul Scholtz, and Nancy Xiao for assistance in data acquisition and analysis. The research described in this paper was performed at the Jet Propulsion Laboratory, California Institute of Technology, under a contract with the National Aeronautics and Space Administration and was supported in part by ARPA under grant number N00014-92-J-1860

## References

R. Patton, P. Frank, and R. Clark (eds.), *Fault Diagnosis in Dynamic Systems: Theory and Application*, New York, NY: Prentice Hall, 1989.

P. Smyth and J. Mellstrom, 'Fault diagnosis of antenna pointing systems using hybrid neural networks and signal processing techniques,' in *Advances in Neural Information Processing Systems 4*, J. E. Moody, S. J. Hanson, R. P. Lippmann (eds.), San Mateo, CA: Morgan Kaufmann, pp.667–674, 1992.

P. Smyth, 'Hidden Markov models for fault detection in dynamic systems,' *Pattern Recognition*, vol.27, no.1, in press.

M. D. Richard and R. P. Lippmann, 'Neural network classifiers estimate Bayesian a posteriori probabilities,' *Neural Computation*, 3(4), pp.461–483, 1992.

J. Miller, R. Goodman, and P. Smyth, 'On loss functions which minimize to conditional expected values and posterior probabilities,' *IEEE Transactions on Information Theory*, vol.39, no.4, pp.1404–1408, July 1993.

P. Smyth, 'Probability density estimation and local basis function neural networks,' in *Computational Learning Theory and Natural Learning Systems*, T. Petsche, M. Kearns, S. Hanson, R. Rivest (eds.), Cambridge, MA: MIT Press, 1994.

P. Smyth and J. Mellstrom, 'Failure detection in dynamic systems: model construction without fault training data,' *Telecommuncations and Data Acquisition Progress Report, vol. 112*, pp.37–49, Jet Propulsion Laboratory, Pasadena, CA, February 15th 1993.

A. Stolcke and S. Omohundro, 'Hidden Markov model induction by Bayesian merging,' in *Advances in Neural Information Processing Systems 5*, C. L. Giles, S. J. Hanson and J. D. Cowan (eds.), San Mateo, CA: Morgan Kaufmann, pp.11–18, 1993.

J. Aitchison, J. D. F. Habbema, and J. W. Kay, 'A critical comparison of two methods of statistical discrimination,' *Applied Statistics*, vol.26, pp.15–25, 1977.

B. Dubuisson and M. Masson, 'A statistical decision rule with incomplete knowledge about the classes,' *Pattern Recognition*, vol.26, no.1, pp.155-165, 1993.